# Global seismic monitoring as probabilistic inference

**Nimar S. Arora**
Department of Computer Science
University of California, Berkeley
Berkeley, CA 94720
nimar@cs.berkeley.edu

**Stuart Russell**
Department of Computer Science
University of California, Berkeley
Berkeley, CA 94720
russell@cs.berkeley.edu

**Paul Kidwell**
Lawrence Livermore National Lab
Livermore, CA 94550
kidwell1@llnl.gov

**Erik Sudderth**
Department of Computer Science
Brown University
Providence, RI 02912
sudderth@cs.brown.edu

## Abstract

The International Monitoring System (IMS) is a global network of sensors whose purpose is to identify potential violations of the Comprehensive Nuclear-Test-Ban Treaty (CTBT), primarily through detection and localization of seismic events. We report on the first stage of a project to improve on the current automated software system with a Bayesian inference system that computes the most likely global event history given the record of local sensor data. The new system, VISA (Vertically Integrated Seismological Analysis), is based on empirically calibrated, generative models of event occurrence, signal propagation, and signal detection. VISA exhibits significantly improved precision and recall compared to the current operational system and is able to detect events that are missed even by the human analysts who post-process the IMS output.

## 1 Introduction

The CTBT aims to prevent the proliferation and the advancement of nuclear weapon technology by banning all nuclear explosions. A global network of seismic, radionuclide, hydroacoustic, and infrasound sensors, the IMS, has been established to enforce the treaty. The IMS is the world's primary global-scale, continuous, real-time system for seismic event monitoring. Data from the IMS sensors are transmitted via satellite in real time to the International Data Center (IDC) in Vienna, where automatic event-bulletins are issued at predefined latency. Perfect performance remains well beyond the reach of current technology: the IDC's automated system, a highly complex and well-tuned piece of software, misses nearly one third of all seismic events in the magnitude range of interest, and about half of the reported events are spurious. A large team of expert analysts post-processes the automatic bulletins to improve their accuracy to acceptable levels.

Like most current systems, the IDC operates by *detection* of arriving signals at each sensor station (the *station processing* stage) and then grouping multiple detections together to form *events* (the *network processing* stage).[1] The time and location of each event are found by various search methods including grid search [2], the double-difference algorithm [3], and the intersection method [4]. In the words of [5], "Seismic event location is—at its core—a minimization of the difference between observed and predicted arrival times." Although the mathematics of seismic event detection and

localization has been studied for almost 100 years [6], the IDC results indicate that the problem is far from trivial.

There are three primary sources of difficulty: 1) the *travel time* between any two points on the earth and the *attenuation* of various frequencies and wave types are not known accurately; 2) each detector is subject to local *noise* that may mask true signals and cause false detections (as much as $90\%$ of all detections are false); and 3) there are many thousands of detections per day, so the combinatorial problem of proposing and comparing possible events (subsets of detections) is daunting. These considerations suggest that an approach based on probabilistic inference and combination of evidence might be effective, and this paper demonstrates that this is in fact the case. For example, such an approach automatically takes into account *non-detections* as negative evidence for a hypothesized event, something that classical methods cannot do.

In simple terms, let $X$ be a random variable ranging over all possible collections of events, with each event defined by time, location, magnitude, and type (natural or man-made). Let $Y$ range over all possible waveform signal recordings at all detection stations. Then $P_\theta(X)$ describes a parameterized generative prior over events, and $P_\phi(Y \mid X)$ describes how the signal is propagated and measured (including travel time, selective absorption and scattering, noise, artifacts, sensor bias, sensor failures, etc.). Given observed recordings $Y = y$, we are interested in the posterior $P(X \mid Y = y)$, and perhaps in the value of $X$ that maximizes it—i.e., the most likely explanation for all the sensor readings. We also learn the model parameters $\theta$ and $\phi$ from historical data.

Our overall project, VISA (Vertically Integrated Seismic Analysis), is divided into two stages. The first stage, NET-VISA, is the subject of the current paper. As the name suggests, NET-VISA deals only with network processing and relies upon the IDC's pre-existing signal detection algorithms. (The second stage, SIG-VISA, will incorporate a signal waveform model and thereby subsume the detection function.) NET-VISA computes a single most-likely explanation: a set of hypothesized events with their associated detections, marking all other detections as noise. This input-output specification, while not fully Bayesian in spirit, enables direct comparison to the current automated system bulletin, SEL3. Using the final expert-generated bulletin, LEB, as ground truth, we compared the two systems on 7 days of held-out data. NET-VISA has $16\%$ more recall at the same precision as SEL3, and $25\%$ more precision at the same recall as SEL3. Furthermore, taking data from the more comprehensive NEIC (National Event Information Center) database as ground truth for the continental United States, we find that NET-VISA is able to detect events in the IMS data that are not in the LEB report produced by IDC's expert analysts; thus, NET-VISA's true performance may be higher than the LEB-based calculation would suggest.

The rest of the paper is structured as follows. Section 2 describes the problem in detail and covers some elementary seismology. Sections 3 and 4 describe the probability model and inference algorithm. Section 5 presents the results of our evaluation, and Section 6 concludes.

## 2    The Seismic Association and Localization Problem

Seismic events are disturbances in the earth's crust. Our work is concerned primarily with earthquakes and explosions (nuclear and conventional), but other types of events—waves breaking, trees falling, ice falling, etc.—may generate seismic waves too. All such waves occur in a variety of types [7]—*body waves* that travel through the earth's interior and *surface waves* that travel on the surface. There are two types of body waves—compression or P waves and shear or S waves. There are also two types of surface waves—Love and Rayleigh. Further, body waves may be reflected off different layers of the earth's crust and these are labeled distinctly by seismologists. Each particular wave type generated by a given event is called a *phase*. These waves are picked up in seismic stations as ground vibrations. Typically, seismic stations have either a single 3-axis detector or an *array* of vertical-axis detectors spread over a scale of many kilometers. Most detectors are sensitive to nanometer-scale displacements, and so are quite susceptible to noise.

Raw seismometer measurements are run through standard signal processing software that filters out non-seismic frequencies and computes short-term and long-term averages of the signal amplitude. When the ratio of these averages exceeds a fixed threshold, a *detection* is announced. Various parameters of the detection are measured—*onset time*, *azimuth* (direction from the station to the source of the wave), *slowness* (related to the angle of declination of the signal path), *amplitude*, etc.

Based on these parameters, a phase label may be assigned to the detection based on the standard IASPEI phase catalog [7]. All of these detection attributes may be erroneous.

The problem that we attempt to solve in this paper is to take a continuous stream of detections (with onset time, azimuth, slowness, amplitude, and phase label) from the roughly 120 IMS seismic stations as input and produce a continuous stream of events and associations between events and detections. The parameters of an event are its longitude, latitude, depth, time, and magnitude ($m_b$ or body-wave magnitude). A 3-month dataset (660 GB) has been made available by the IDC for the purposes of this research. We have divided the dataset into 7 days of validation, 7 days of test, and the rest as training data. We compute the accuracy of an event history hypothesis by comparison to a chosen ground-truth history. A bipartite graph is created between predicted and true events. An edge is added between a predicted and a true event that are at most 5 degrees in distance[2] and 50 seconds in time apart. The weight of the edge is the distance between the two events. Finally, a min-weight max-cardinality matching is computed on the graph. We report 3 quantities from this matching— *precision* (percentage of predicted events that are matched), *recall* (percentage of true events that are matched), and *average error* (average distance in kilometers between matched events).

## 3   Generative Probabilistic Model

Our generative model for seismic events and detections follows along the lines of the aircraft detection model in [8, Figure 3]. In our model, there is an unknown number of seismic events with unknown parameters (location, time, etc.). These events produce 14 different types of seismic waves or phases. A phase from an event may or may not be detected by a station. If a phase is detected at a station, a corresponding detection is generated. However, the parameters of the detection may be imprecise. Additionally, an unknown number of noise detections are generated at each station. For NET-VISA, the evidence $Y = y$ consists only of each station's set of detections and their parameters.

### 3.1   Events

The events are generated by a time-homogeneous Poisson process. If $e$ is the set of events (of size $|e|$), $\lambda_e$ is the rate of event generation, and $T$ is the time period under consideration, we have

$$P_\theta(|e|) = \frac{(\lambda_e \cdot T)^{|e|} \exp\left(-\lambda_e \cdot T\right)}{|e|!} \; . \tag{1}$$

The longitude and latitude of the $i$th event, $e_l^i$ are drawn from an event location density, $p_l(e_l)$ on the surface of the earth. The depth of the event, $e_d^i$ is uniformly distributed up to a maximum depth $D$ (700 km in our experiments). Similarly, the time of the event $e_t^i$ is uniformly distributed between 0 and $T$. The magnitude of the event, $e_m^i$, is drawn from what seismologists refer to as the Gutenberg-Richter distribution, which is in fact an exponential distribution with rate $\lambda_m$:

$$P_\theta(e^i) = p_l(e_l^i)\frac{1}{D}\frac{1}{T}\lambda_m \exp\left(-\lambda_m e_m^i\right) \; . \tag{2}$$

Since all the events are exchangeable, we have

$$P_\theta(e) = P_\theta(|e|) \cdot |e|! \cdot \prod_{i=1}^{|e|} P_\theta(e^i) = \exp\left(-\lambda_e \cdot T\right) \prod_{i=1}^{|e|} p_l(e_l^i)\frac{1}{D}\lambda_e\lambda_m \exp\left(-\lambda_m e_m^i\right) \; . \tag{3}$$

Maximum likelihood estimates of $\lambda_e$ and $\lambda_m$ may be easily determined from historical event frequencies and magnitudes. To approximate $p_l(e_l)$, we use a kernel density estimate derived from the following exponentially decaying kernel:

$$K_{b,x}(y) = \frac{1 + 1/b^2}{2\pi R^2} \frac{\exp\left(-\Delta_{xy}/b\right)}{1 + \exp\left(-\pi/b\right)} \; . \tag{4}$$

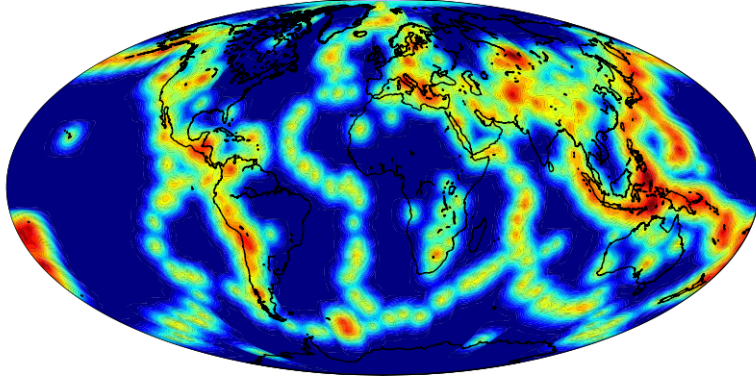

Figure 1: Heat map (large values in red, small in blue) of the prior event location density $\log p_l(e_l)$.

Here $b > 0$ is the bandwidth, $\Delta_{xy}$ is the distance (in radians) between locations $x$ and $y$ on the surface of the earth, and $R$ is the earth's radius. The bandwidth was estimated via cross-validation. In addition, we additively mixed this kernel density with a uniform distribution, with prior probability 0.001, to allow the possibility of explosions at an arbitrary location. The overall density, as illustrated in Figure 1, was pre-computed on a one degree grid and interpolated during inference.

### 3.2 Correct Detections

The probability that an event's $j^{th}$ phase, $1 \leq j \leq J$, is detected by a station $k$, $1 \leq k \leq K$, depends on the wave type or phase, the station, and the event's magnitude, depth, and distance to the station. Let $d^{ijk}$ be a binary indicator variable for such a detection of event $i$, and $\Delta_{ik}$ the distance between event $i$ and station $k$. Then we have

$$P_\phi(d^{ijk} = 1 \mid e^i) = p_d^{jk}(e_m^i, e_d^i, \Delta_{ik}). \tag{5}$$

If an event phase is detected at a station, i.e. $d^{ijk} = 1$, our model specifies probability distribution for the attributes of that detection, $a^{ijk}$. The arrival time, $a_t^{ijk}$, is assigned a Laplacian distribution whose mean consists of two parts. The first is the IASPEI travel time prediction for that phase, which depends only on the event depth and the distance between the event and station. The second is a learned station-specific correction which accounts for inhomogeneities in the earth's crust, which allow seismic waves to travel faster or slower than the IASPEI prediction. The station-specific correction also accounts for any systematic biases in picking seismic onsets from waveforms. Let $\mu_t^{jk}$ be the location of this Laplacian (a function of the event time, depth, and distance to the station) and let $b_t^{jk}$ be its scale. Truncating this Laplacian to the range of possible arrival times produces a normalization constant $Z_t^{jk}$, so that

$$P_\phi(a_t^{ijk} \mid d^{ijk} = 1, e^i) = \frac{1}{Z_t^{jk}} \exp\left(-\frac{|a_t^{ijk} - \mu_t^{jk}(e_t^i, e_d^i, \Delta_{ik})|}{b_t^{jk}}\right). \tag{6}$$

Similarly, the arrival azimuth and slowness follow a Laplacian distribution. The location $a_z^{ijk}$ of the arrival azimuth depends only on the location of the event, while the location $a_s^{ijk}$ of the arrival slowness depends only on the event depth and distance to the station. The scales of all these Laplacians are fixed for a given phase and station, so that

$$P_\phi(a_z^{ijk} \mid d^{ijk} = 1, e^i) = \frac{1}{Z_z^{jk}} \exp\left(-\frac{|a_z^{ijk} - \mu_z^{jk}(e_l^i)|}{b_z^{jk}}\right), \tag{7}$$

$$P_\phi(a_s^{ijk} \mid d^{ijk} = 1, e^i) = \frac{1}{Z_s^{jk}} \exp\left(-\frac{|a_s^{ijk} - \mu_s^{jk}(e_d^i, \Delta_{ik})|}{b_s^{jk}}\right). \tag{8}$$

The arrival amplitud $a_a^{ijk}$ is similar to the detection probability in that it depends only on the event magnitude, depth, and distance to the station. We model the log of the amplitude via a linear regression model with Gaussian noise:

$$P_\phi(a_a^{ijk} \mid d^{ijk} = 1, e^i) = \frac{1}{\sqrt{2\pi}\sigma_a^{jk}} \exp\left(-\frac{(log(a_a^{ijk}) - \mu_a^{jk}(e_m^i, e_d^i, \Delta_{ik}))^2}{2\sigma_a^{jk^2}}\right). \tag{9}$$

Finally, the phase label $a_h^{ijk}$ automatically assigned to the detection follows a multinomial distribution whose parameters depends on the true phase, $j$:

$$P_\phi(a_h^{ijk} \mid d^{ijk} = 1, e^i) = p_h^{jk}(a_h^{ijk}). \tag{10}$$

The phase- and station-specific detection distributions, $p_d^{jk}(\cdot)$, were obtained using logistic regression models estimated via a hierarchical Bayesian procedure [9]. Because phase labels indicate among other things the general physical path taken from an event to a station, a distinct set of features were learned from the event characteristics for each phase. To estimate the individual station weights $\alpha_{wjk}$ for each phase $j$ and feature $w$, a hierarchical model was specified in which each station-specific weight is independently drawn from a feature-dependent global Normal distribution, so that $\alpha_{wjk} \sim N(\mu_{wj}, \sigma_{wj}^2)$. Weakly informative diffuse priors $\mu_{wj} \sim N(0, 100^2)$, $\sigma_{wj}^{-2} \sim \text{Gamma}(0.01, 0.01)$, were placed on the parameters of these global distributions, and posterior mean estimates of the station-specific weights obtained via Gibbs sampling. Figure 2 shows two of the empirical and modeled distributions for one phase-site.

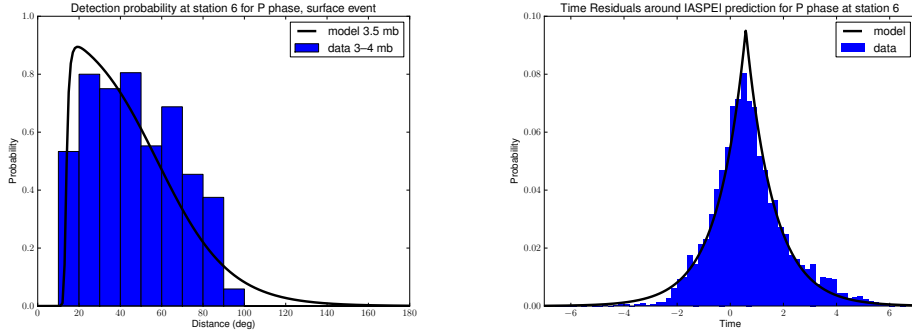

Figure 2: Conditional detection probabilities and arrival time distributions (relative to the IASPEI prediction) for the P phase at Station 6.

## 3.3 False Detections

Each station, $k$, also generates a set of false detections $f^k$ through a time-homogeneous Poisson process with rate $\lambda_f^k$:

$$P_\phi(|f^k|) = \frac{(\lambda_f^k \cdot T)^{|f^k|} \exp\left(-\lambda_f^k \cdot T\right)}{|f^k|!}. \tag{11}$$

The time $f_t^{kl}$, azimuth $f_z^{kl}$, and slowness $f_s^{kl}$ of these false detections are generated uniformly over their respective ranges. The amplitude $f_a^{kl}$ of the false detection is generated from a mixture of two Gaussians, $p_a^k(f_a^{kl})$. Finally, the phase label $f_h^{kl}$ assigned to the false detection follows a multinomial distribution, $p_h^k(f_h^{kl})$. If the azimuth and slowness take values on ranges of length $M_z$ and $M_s$, respectively, then the probability of the $l^{th}$ false detection is given by

$$P_\phi(f^{kl}) = \frac{1}{T}\frac{1}{M_z}\frac{1}{M_s} p_a^k(f_a^{kl}) p_h^k(f_h^{kl}). \tag{12}$$

Since the false detections at a station are exchangeable, we have

$$P_\phi(f^k) = P_\phi(|f^k|) \cdot |f^k|! \prod_{l=1}^{l=|f^k|} P_\phi(f^{kl}) = \exp\left(-\lambda_f^k \cdot T\right) \prod_{l=1}^{l=|f^k|} \frac{\lambda_f^k}{M_z M_s} p_a^k(f_a^{kl}) p_h^k(f_h^{kl}). \tag{13}$$

# 4 Inference

Combining the model components developed in the preceding section, the overall probability of any hypothesized sequence of events $e$, detected event phases $d$, arrival attributes $a$ for correctly detected event phases, and arrival attributes $f$ for falsely detected events is

$$P(e, d, a, f) = P_\theta(e) P_\phi(d \mid e) P_\phi(a \mid d, e) P_\phi(f). \tag{14}$$

We will attempt to find the most likely explanation consistent with the observations. This involves determining $e$, $d$, $a$, and $f$ which maximize $P(e, d, a, f)$, such that the set of detections implied by $d$, $a$, and $f$ correspond exactly with the observed detections. Since detections from real seismic sensors are observed incrementally and roughly in time-ascending order, our inference algorithm also produces an incremental hypothesis which advances with time. Our algorithm can be seen as a form of greedy search, in which the current hypothesis is improved via a set of local moves.

Let $M_T$ denote the maximum travel time for any phase. Initially, we start with an event-window of size $W$ from $t_0 = 0$ to $t_1 = W$, and a detection-window of size $W + M_T$ from $t_0 = 0$ to $t_1 = W + M_T$. Our starting hypothesis is that all detections in our detection-window are false detections and there are no events. We then repeatedly apply the birth, death, improve-event, and improve-detection moves (described below) for a fixed number of iterations ($N$ times the number of detections in that window) before shifting the windows forward by a step size $S$. Any new detections added to the detection window are again assumed to be false detections. As the windows move forward the events older than $t_0 - M_T$ become stable: none of the moves modify either the event or detections associated with them. These events are then output. While in theory this algorithm never needs to terminate, our experiments continue until the test dataset is fully consumed.

In order to simplify the computations needed to compare alternate hypotheses, we decompose the overall probability of Eq. (14) into the contribution from each event. We define the score $S_e$ of an event as the probability ratio of two hypotheses: one in which the event exists, and another in which the event doesn't exist and all of its associated detections are noise. If an event has score less than 1, an alternative hypothesis in which the event is deleted clearly has higher probability. Critically, this event score is unaffected by other events in the current hypothesis. From Eqs. (3) and (13) we have

$$S_e(e^i) = \frac{p_l(e_l^i)\lambda_e\lambda_m}{D} \exp\left(-\lambda_m e_m^i\right) \prod_{j,k} P_\phi(d^{ijk} \mid e^i) \left(\delta(d^{ijk}, 0) + \delta(d^{ijk}, 1) \frac{P_\phi(a^{ijk} \mid d^{ijk}, e^i)}{\frac{\lambda_f^k}{M_z M_s} p_a^k(f_a^{kl}) p_h^k(f_h^{kl})}\right).$$

Note that the final fraction is a likelihood ratio comparing interpretations of the same detection as either the detection of event $i$'s $j^{th}$ phase at station $k$, or the $l^{th}$ false detection at station $k$. We can further decompose the score into scores $S_d$ for each detection. The score of $d^{ijk}$, defined when $d^{ijk} = 1$, is the ratio of the probabilities of the hypothesis where the detection is associated with phase $j$ of event $i$ at station $k$, and one in which this detection is false and phase $j$ of event $i$ is missed by station $k$:

$$S_d(d^{ijk}) = \frac{p_d^{jk}(e_m^i, e_d^i, \Delta_{ik})}{1 - p_d^{jk}(e_m^i, e_d^i, \Delta_{ik})} \frac{P_\phi(a^{ijk} \mid d^{ijk}, e^i)}{\frac{\lambda_f^k}{M_z M_s} p_a^k(f_a^{kl}) p_h^k(f_h^{kl})}. \tag{15}$$

By definition, any detection with score less than 1 is more likely to be a false detection. Also, the score of an individual detection is independent of other detections and unassociated events in the hypothesis. These scores play a key role in the following local search moves.

**Birth Move** We randomly pick a detection, invert it into an event location (using the detection's time, azimuth, and slowness), and sample an event in a 10 degree by 100 second ball around this inverted location. The depth of the event is fixed at 0, and the magnitude is uniformly sampled.

**Improve Detections Move** For each detection in the detection window, we consider all possible phases $j$ of all events $i$ up to $M_T$ seconds earlier. We then associate the best event-phase for this detection that is not already assigned to a detection with higher score at the same station $k$. If this best event-phase has score $S_d(d^{ijk}) < 1$, the detection is changed to a false detection.

**Improve Events Move** For each event $e^i$, we consider 10 points chosen uniformly at random in a small ball around the event (2 degrees in longitude and latitude, 100 km in depth, 5 seconds in time, and 2 units of magnitude), and choose those attributes with the highest score $S_e(e^i)$.

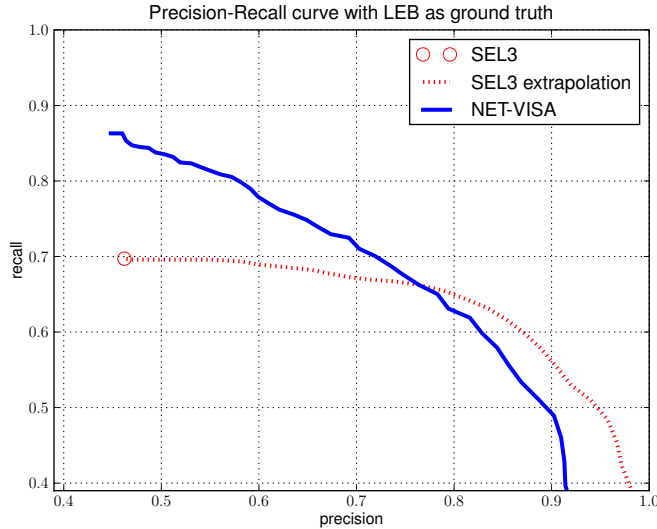

Figure 3: Precision-recall performance of the proposed NET-VISA and deployed SEL3 algorithms, treating the analyst-generated LEB as ground truth.

**Death Move**   Any event $e^i$ with score $S_e(e^i) < 1$ is deleted, and all of its currently associated detections are marked as false alarms.

**Final Pruning**   Before outputting event hypotheses, we perform a final round of pruning to remove some duplicate events. In particular, we delete any event for which there is another higher-scoring event within 5 degrees distance and 50 seconds time. Such spurious, or *shadow*, event hypotheses arise because real seismic events generate many more phases than we currently model. In addition, a single phase may sometimes generate multiple detections due to waveform processing, or "pick", errors. These additional unmodeled detections, when taken together, often suggest an additional event at about the same location and time as the original event.

Note that the birth move is not a greedy move: the proposed event will almost always have a score $S_e(e^i) < 1$ until some number of detections are assigned in subsequent moves. The overall structure of these moves could be easily converted to an MCMC or simulated annealing algorithm. However, in our experiments this search outperformed simple MCMC methods in terms of speed and accuracy.

## 5   Experimental Results

As discussed in Section 2, we measure the precision, recall, and average error of our predictions via an assumed ground truth. We first treat the IMS analyst-generated LEB as ground truth, and compare the performance of our NET-VISA algorithm to the currently deployed SEL3 system. Using the scores for hypothesized events, we have generated a precision-recall curve for NET-VISA, and marked SEL3 on it as a point (see Figure 3). Also in this figure, we show a precision-recall curve for SEL3 using scores from an SVM trained to classify true and false SEL3 events [10] (SEL3 extrapolation). As shown in the figure, NET-VISA has at least 16% more recall at the same precision as SEL3, and at least 25% more precision at the same recall as SEL3.

The true precision of NET-VISA is perhaps higher than this comparison suggests. We have evaluated the recall of LEB and NET-VISA with the NEIC dataset as ground truth. Since the NEIC has many more sensors in the United States than the IMS, it is considered a more reliable summary of seismic activity in this region. Out of 33 events in the continental United States, LEB found 4, and NET-VISA found 8 including the 4 found by LEB.

Figure 4 shows the recall and error divided among different types of LEB events. The table on the left shows a break-down by LEB event magnitude. For magnitudes up to 4, NET-VISA has nearly 20% higher recall with similar error. The table on the right shows a break-down by *azimuth*

| $m_b$ | Count | SEL3 Recall | SEL3 Err | NET-VISA Recall | NET-VISA Err | Azimuth Gap | Count | SEL3 Recall | SEL3 Err | NET-VISA Recall | NET-VISA Err |
|---|---|---|---|---|---|---|---|---|---|---|---|
| $0-2$ | 74 | 64.9 | 101 | 85.1 | 91 | $0-90$ | 72 | 100.0 | 28 | 100.0 | 38 |
| $2-3$ | 36 | 50.0 | 186 | 75.0 | 171 | $90-180$ | 315 | 88.9 | 76 | 93.7 | 72 |
| $3-4$ | 558 | 66.5 | 104 | 85.1 | 109 | $180-270$ | 302 | 51.0 | 134 | 82.1 | 126 |
| $>4$ | 164 | 86.6 | 70 | 93.3 | 80 | $270-360$ | 143 | 51.0 | 176 | 72.0 | 187 |
| all | 832 | 69.7 | 99 | 86.3 | 103 | all | 832 | 69.7 | 99 | 86.3 | 103 |

Figure 4: Recall and error (km) broken down by LEB event magnitude and azimuth gap (degrees).

*gap*, defined as the largest difference in consecutive event-to-station azimuths for stations which detect an event. Large gaps indicate that the event location is under-constrained. For example, if all stations are to the southwest of an event, the gap is greater than 270 degrees and the event will be poorly localized along a line running from southwest to northeast. By using evidence about missed detections ignored by SEL3, NET-VISA reduces this uncertainty and performs much better.

All of the results in this section were produced using 7 days of data from the validation set. The inference used a window size, $W$, of 30 minutes, a step size, $S$, of 15 minutes, and $N = 1000$ iterations. There were a total of 832 LEB events during this period, and roughly 120,000 detections. The inference took about 4.5 days on a single core running at 2.5 GHz. Estimating model parameters from 2.5 months of training data took about 1 hour.

## 6   Conclusions and Further Work

Our results demonstrate that a Bayesian approach to seismic monitoring can improve significantly on the performance of classical systems. The NET-VISA system can not only reduce the human analyst effort required to achieve a given level of accuracy, but can also lower the magnitude threshold for reliable detection. Given that the difficulty of seismic monitoring was cited as one of the principal reasons for non-ratification of the CTBT by the United States Senate in 1999, one hopes that improvements in monitoring may increase the chances of final ratification and entry into force.

Putting monitoring onto a sound probabilistic footing also facilitates further improvements such as continuous estimation of local noise conditions, travel time, and attenuation models without the need for ground-truth calibration experiments (controlled explosions). We also expect to lower the detection threshold significantly by extending the generative model to include waveform characteristics, so that detection becomes part of a globally integrated inference process—and hence susceptible to top-down influences—rather than being a purely local, bottom-up, hard-threshold decision.

**Acknowledgments**

We would like to thank the many seismologists who patiently explained to us the intricacies of their field, among them Ronan LeBras, Robert Engdahl, David Bowers, Bob Pearce, Stephen Myers, Dmitry Storchak, Istvan Bondar, and Barbara Romanowicz. We also received assistance from several Berkeley undergraduates, including Matthew Cann, Hong Hu, Christopher Lin, and Andrew Lee. The third author's work was performed under the auspices of the U.S. Department of Energy at Lawrence Livermore National Laboratory under Contract DE-AC52-07NA27344. The other authors were partially supported by the Preparatory Commission for the CTBT. Finally, the first author wishes to thank his family for their infinite patience and support.

## Footnotes

[1]Network processing is thus a *data association* problem similar to those arising in multitarget tracking [1].

[2]In this paper, by distance between two points on the surface of the earth we refer to the great-circle distance. This can be represented in degrees, radians, or kilometers (using the average earth radius of 6371 km).

## References

[1] Y. Bar-Shalom and T.E. Fortmann. *Tracking and Data Association*. Academic Press, 1988.

[2] P. M. Shearer. Improving local earthquake location using the L1 norm and waveform cross correlation: Application to the Whittier Narrows, California, aftershock sequence. *J. Geophys. Res.*, 102:8269 – 8283, 1997.

[3] F. Waldhauser and W. L. Ellsworth. A double-difference earthquake location algorithm: method and application to the Northern Hayward Fault, California. *Bulletin of the Seismological Society of America*, 90:1353 – 1368, 2000.

[4] J. Pujol. Earthquake location tutorial: graphical approach and approximate epicentral location techniques. *Seis. Res. Letter*, 75:63 – 74, 2004.

[5] Stephen C. Myers, Gardar Johannesson, and William Hanley. A Bayesian hierarchical method for multiple-event seismic location. *Geophysical Journal International*, 171:1049–1063, 2009.

[6] L. Geiger. Probability method for the determination of earthquake epicenters from the arrival time only. *Bull. St. Louis Univ.*, 8:60 –71, 1912.

[7] D. A. Storchak, J. Schweitzer, and P. Bormann. The IASPEI standard seismic phase list. *Seismol. Res. Lett.*, 74(6):761 – 772, 2003.

[8] Brian Milch, Bhaskara Marthi, Stuart J. Russell, David Sontag, Daniel L. Ong, and Andrey Kolobov. BLOG: Probabilistic models with unknown objects. In *IJCAI*, pages 1352–1359, 2005.

[9] A. Gelman, J. B. Carlin, H. S. Stern, and D. B. Rubin. *Bayesian Data Analysis*. Chapman & Hall, 2004.

[10] Lester Mackey, Ariel Kleiner, and Michael I. Jordan. Improved automated seismic event extraction using machine learning. *Eos Trans. AGU*, 90(52), 2009. Fall Meet. Suppl., Abstract S31B-1714.

